# Learning models of object structure

**Joseph Schlecht**
Department of Computer Science
University of Arizona
schlecht@cs.arizona.edu

**Kobus Barnard**
Department of Computer Science
University of Arizona
kobus@cs.arizona.edu

## Abstract

We present an approach for learning stochastic geometric models of object categories from single view images. We focus here on models expressible as a spatially contiguous assemblage of blocks. Model topologies are learned across groups of images, and one or more such topologies is linked to an object category (e.g. chairs). Fitting learned topologies to an image can be used to identify the object class, as well as detail its geometry. The latter goes beyond labeling objects, as it provides the geometric structure of particular instances. We learn the models using joint statistical inference over category parameters, camera parameters, and instance parameters. These produce an image likelihood through a statistical imaging model. We use trans-dimensional sampling to explore topology hypotheses, and alternate between Metropolis-Hastings and stochastic dynamics to explore instance parameters. Experiments on images of furniture objects such as tables and chairs suggest that this is an effective approach for learning models that encode simple representations of category geometry and the statistics thereof, and support inferring both category and geometry on held out single view images.

## 1 Introduction

In this paper we develop an approach to learn stochastic 3D geometric models of object categories from single view images. Exploiting such models for object recognition systems enables going beyond simple labeling. In particular, fitting such models opens up opportunities to reason about function or utility, how the particular object integrates into the scene (i.e., perhaps it is an obstacle), how the form of the particular instance is related to others in its category (i.e., perhaps it is a particularly tall and narrow one), and how categories themselves are related.

Capturing the wide variation in both topology and geometry within object categories, and finding good estimates for the underlying statistics, suggests a large scale learning approach. We propose exploiting the growing number of labeled single-view images to learn such models. While our approach is trivially extendable to exploit multiple views of the same object, large quantities of such data is rare. Further, the key issue is to learn about the variation of the category. Put differently, if we are limited to 100 images, we would prefer to have 100 images of different examples, rather than, say, 10 views of 10 examples.

Representing, learning, and using object statistical geometric properties is potentially simpler in the context of 3D models. In contrast, statistical models that encode image-based appearance characteristics and/or part configuration statistics must deal with confounds due to the imaging process. For example, right angles in 3D can have a wide variety of angles in the image plane, leading to using the same representations for both structure variation and pose variation. This means that the represented geometry is less specific and less informative. By contrast, encoding the structure variation in 3D models is simpler and more informative because they are linked to the object alone.

To deal with the effect of an unknown camera, we estimate the camera parameters simultaneously while fitting the model hypothesis. A 3D model hypothesis is a relatively strong hint as to what

the camera might be. Further, we make the observation that the variations due to standard camera projection are quite unlike typical category variation. Hence, in the context of a given object model hypothesis, the fact that the camera is not known is not a significant impediment, and much can be estimated about the camera under that hypothesis.

We develop our approach with object models that are expressible as a spatially contiguous assemblage of blocks. We include in the model a constraint on right angles between blocks. We further simplify matters by considering images where there are minimal distracting features in the background. We experiment with images from five categories of furniture objects. Within this domain, we are able to automatically learn topologies. The models can then be used to identify the object category using statistical inference. Recognition of objects in clutter is likely effective with this approach, but we have yet to integrate support for occlusion of object parts into our inference process.

We learn the parameters of each category model using Bayesian inference over multiple image examples for the category. Thus we have a number of parameters specifying the category topology that apply to all images of objects from the category. Further, as a side effect, the inference process finds instance parameters that apply specifically to each object. For example, all tables have legs and a top, but the proportions of the parts differ among the instances. In addition, the camera parameters for each image are determined, as these are simultaneously fit with the object models. The object and camera hypotheses are combined with an imaging model to provide the image likelihood that drives the inference process.

For learning we need to find parameters that give a high likelihood of the data from multiple examples. Because we are searching for model topologies, we need to search among models with varying dimension. For this we use the trans-dimensional sampling framework [7, 8]. We explore the posterior space within a given probability space of a particular dimension by combining standard Metropolis-Hastings [1, 14], with stochastic dynamics [18]. As developed further below, these two methods have complementary strengths for our problem. Importantly, we arrange the sampling so that the hybrid of samplers are guaranteed to converge to the posterior distribution. This ensures that the space will be completely explored, given enough time.

**Related work.** Most work on learning representations for object categories has focused on image-based appearance characteristics and/or part configuration statistics (e.g., [4, 5, 6, 12, 13, 24]). These approaches typically rely on effective descriptors that are somewhat resilient to pose change (e.g., [16]). A second force favoring learning 2D representations is the explosion of readily available images compared with that for 3D structure, and thus treating category learning as statistical pattern recognition is more convenient in the data domain (2D images). However, some researchers have started imposing more projective geometry into the spatial models. For example, Savarese and Fei-Fei [19, 20] build a model where arranged parts are linked by a fundamental matrix. Their training process is helped by multiple examples of the same objects, but notably they are able to use training data with clutter. Their approach is different than ours in that models are built more bottom up, and this process is somewhat reliant on the presence of surface textures. A different strategy proposed by Hoeim et al. [9] is to fit a deformable 3D blob to cars, driven largely by appearance cues mapped onto the model. Our work also relates to recent efforts in learning abstract topologies [11, 26] and structure models for 2D images of objects constrained by grammar representations [29, 30]. Also relevant is a large body of older work on representing objects with 3D parts [2, 3, 28] and detecting objects in images given a precise 3D model [10, 15, 25], such as one for machined parts in an industrial setting. Finally, we have also been inspired by work on fitting deformable models of known topology to 2D images in the case of human pose estimation (e.g., [17, 22, 23]).

## 2   Modeling object category structure

We use a generative model for image features corresponding to examples from object categories (Fig. 1). A category is associated with a sampling from category level parameters which are the number of parts, $\mathbf{n}$, their interconnections (topology), $\mathbf{t}$, the structure statistics $\mathbf{r_s}$, and the camera statistics, $\mathbf{r_s}$. Associating camera distributional parameters with a category allows us to exploit regularity in how different objects are photographed during learning. We support clusters within categories to model multiple structural possibilities (e.g., chairs with and without arm rests). The cluster variable, $\mathbf{z}$, selects a category topology and structure distributional parameters for attachment locations and part sizes. We denote the specific values for a particular example by $\mathbf{s}$. Similarly, we

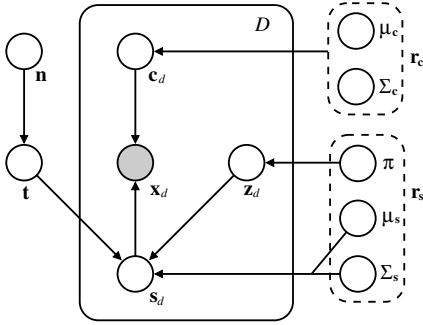

Figure 1: Graphical model for the generative approach to images of objects from categories described by stochastic geometric models. The category level parameters are the number of parts, $\mathbf{n}$, their interconnections (topology), $\mathbf{t}$, the structure statistics $\mathbf{r_s}$, and the camera statistics, $\mathbf{r_s}$. Hyperparameters for category level parameters are omitted for clarity. A sample of category level parameters provides a statistical model for a given category, which is then sampled for the camera and object structure values $\mathbf{c}_d$ and $\mathbf{s}_d$, optionally selected from a cluster within the category by $\mathbf{z}_d$. $\mathbf{c}_d$ and $\mathbf{s}_d$ yield a distribution over image features $\mathbf{x}_d$.

denote the camera capturing it by $\mathbf{c}$. The projected model image then generates image features, $\mathbf{x}$, for which we use edge points and surface pixels. In summary, the parameters for an image are $\boldsymbol{\theta}^{(\mathbf{n})} = (\mathbf{c}, \mathbf{s}, \mathbf{t}, \mathbf{r_c}, \mathbf{r_s}, \mathbf{n})$.

Given a set of $D$ images containing examples of an object category, our goal is to learn the model $\boldsymbol{\Theta}^{(\mathbf{n})}$ generating them from detected features sets $\mathbf{X} = \mathbf{x}_1, \ldots, \mathbf{x}_D$. In addition to category-level parameters shared across instances which is of most interest, $\boldsymbol{\Theta}^{(\mathbf{n})}$ comprises camera models $\mathbf{C} = \mathbf{c}_1, \ldots, \mathbf{c}_D$ and structure part parameters $\mathbf{S} = \mathbf{s}_1, \ldots, \mathbf{s}_D$ assuming a hard cluster assignment. In other words, the camera and the geometry of the training examples are fit collaterally.

We separate the joint density into a likelihood and prior

$$p\left(\mathbf{X}, \boldsymbol{\Theta}^{(\mathbf{n})}\right) = p^{(\mathbf{n})}(\mathbf{X}, \mathbf{C}, \mathbf{S} \,|\, \mathbf{t}, \mathbf{r_c}, \mathbf{r_s})\, p^{(\mathbf{n})}(\mathbf{t}, \mathbf{r_c}, \mathbf{r_s}, \mathbf{n}) , \qquad (1)$$

where we use the notation $p^{(\mathbf{n})}(\cdot)$ for a density function corresponding to $\mathbf{n}$ parts. Conditioned on the category parameters, we assume that the $D$ sets of image features and instance parameters are independent, giving

$$p^{(\mathbf{n})}(\mathbf{X}, \mathbf{C}, \mathbf{S} \,|\, \mathbf{t}, \mathbf{r_c}, \mathbf{r_s}) = \prod_{d=1}^{D} p^{(\mathbf{n})}(\mathbf{x}_d, \mathbf{c}_d, \mathbf{s}_d \,|\, \mathbf{t}, \mathbf{r_c}, \mathbf{r_s}) . \qquad (2)$$

The feature data and structure parameters are generated by a sub-category cluster with weights and distributions defined by $\mathbf{r_s} = (\boldsymbol{\pi}, \boldsymbol{\mu_s}, \boldsymbol{\Sigma_s})$. As previously mentioned, the camera is shared across clusters, and drawn from a distribution defined by $\mathbf{r_c} = (\boldsymbol{\mu_c}, \boldsymbol{\Sigma_c})$. We formalize the likelihood of an object, camera, and image features under $M$ clusters as

$$p^{(\mathbf{n})}(\mathbf{x}_d, \mathbf{c}_d, \mathbf{s}_d \,|\, \mathbf{t}, \mathbf{r_c}, \mathbf{r_s})$$
$$= \sum_{m=1}^{M} \pi_m \underbrace{p^{(n_m)}(\mathbf{x}_d \,|\, \mathbf{c}_d, \mathbf{s}_{md})}_{\text{Image}} \underbrace{p(\mathbf{c}_d \,|\, \boldsymbol{\mu_c}, \boldsymbol{\Sigma_c})}_{\text{Camera}} \underbrace{p^{(n_m)}(\mathbf{s}_{md} \,|\, \mathbf{t}_m, \boldsymbol{\mu}_{\mathbf{s}m}, \boldsymbol{\Sigma}_{\mathbf{s}m})}_{\text{Object}} . \qquad (3)$$

We arrive at equation (3) by introducing a binary assignment vector $\mathbf{z}$ for each image feature set, such that $z_m = 1$ if the $m^{\text{th}}$ cluster generated it and 0 otherwise. The cluster weights are then given by $\pi_m = p(z_m = 1)$.

For the prior probability distribution, we assume category parameter independence, with the clustered topologies conditionally independent given the number of parts. The prior in (1) becomes

$$p^{(\mathbf{n})}(\mathbf{t}, \mathbf{r_c}, \mathbf{r_s}, \mathbf{n}) = p(\mathbf{r_c}) \prod_{m=1}^{M} p^{(n_m)}(\mathbf{t}_m \,|\, n_m)\, p^{(n_m)}(\mathbf{r}_{\mathbf{s}m})\, p(n_m) . \qquad (4)$$

For category parameters in the camera and structure models, $\mathbf{r_c}$ and $\mathbf{r_s}$, we use Gaussian statistics with weak Gamma priors that are empirically chosen. We set the number of parts in the object sub-categories, $\mathbf{n}$ to be geometrically distributed. We set the prior over edges in the topology given $\mathbf{n}$ to be uniform.

## 2.1 Object model

We model object structure as a set of connected three-dimensional block constructs representing object parts. We account for symmetric structure in an object category, e.g., legs of a table or chair,

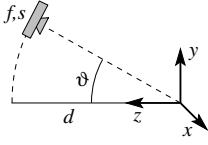 Figure 2: The camera model is constrained to reduce the ambiguity introduced in learning from a single view of an object. We position the camera at a fixed distance and direct its focus at the origin; rotation is allowed about the $x$-axis. Since the object model is allowed to move about the scene and rotate, this model is capable of capturing most images of a scene.

by introducing compound block constructs. We define two constructs for symmetrically aligned pairs (2) or quartets (4) of blocks. Unless otherwise specified, we will use blocks to specify both simple blocks and compound blocks as they handled similarly.

The connections between blocks are made at a point on adjacent, parallel faces. We consider the organization of these connections as a graph defining the structural topology of an object category, where the nodes in the graph represent structural parts and the edges give the connections. We use directed edges, inducing attachment dependence among parts.

Each block has three internal parameters representing its width, height, and length. Blocks representing symmetric pairs or quartets have one or two extra parameters defining the relative positioning of the sub-blocks Blocks potentially have two external attachment parameters $u, v$ where one other is connected. We further constrain blocks to attach to at most one other block, giving a directed tree for the topology and enabling conditional independence among attachments. Note that blocks can be visually "attached" to additional blocks that they abut, but representing them as true attachments makes the model more complex and is not necessary. Intuitively, the model is much like physically building a piece of furniture block by block, but saving on glue by only connecting an added block to one other block. Despite its simplicity, this model can approximate a surprising range of man made objects.

For a set of $n$ connected blocks of the form $\mathbf{b} = (w, h, l, u_1, v_1, \ldots)$, the structure model is $\mathbf{s} = (\varphi, p_o, \mathbf{b}_1, \ldots, \mathbf{b}_n)$. We position the connected blocks in an object coordinate system defined by a point $p_o \in \mathbb{R}^3$ on one of the blocks and a $y$-axis rotation angle, $\varphi$, about this position. Since we constrain the blocks to be connected at right angles on parallel faces, the position of other blocks within the object coordinate system is entirely defined by $p_o$ and the attachments points between blocks.

The object structure instance parameters are assumed Gaussian distributed according to $\boldsymbol{\mu_s}, \boldsymbol{\Sigma_s}$ in the likelihood (3). Since the instance parameters in the object model are conditionally independent given the category, the covariance matrix is diagonal. Finally, for a block $\mathbf{b}_i$ attaching to $\mathbf{b}_j$ on faces defined by the $k^{\text{th}}$ size parameter, the topology edge set is defined as $\mathbf{t} = \left( i, j, k : \mathbf{b}_i \xleftarrow{k} \mathbf{b}_j \right)$.

## 2.2 Camera model

A full specification of the camera and the object position, pose, and scale leads to a redundant set of parameters. We choose a minimal set for inference that retains full expressiveness as follows. Since we are unable to distinguish the actual size of an object from its distance to the camera, we constrain the camera to be at a fixed distance from the world origin. We reduce potential ambiguity from objects of interest being variably positioned in $\mathbb{R}^3$ by constraining the camera to always look at the world origin. Because we allow an object to rotate around its vertical axis, we only need to specify the camera zenith angle, $\vartheta$. Thus we set the horizontal $x$-coordinate of the camera in the world to zero and allow $\vartheta$ to be the only variable extrinsic parameter. In other words, the position of the camera is constrained to a circular arc on the $y, z$-plane (Figure 2). We model the amount of perspective in the image from the camera by parameterizing its focal length, $f$. Our camera instance parameters are thus $\mathbf{c} = (\vartheta, f, s)$, where $\vartheta \in [-\pi/2, \pi/2]$, and $f, s > 0$. The camera instance parameters in (3) are modeled as Gaussian with category parameters $\boldsymbol{\mu_s}, \boldsymbol{\Sigma_s}$.

## 2.3 Image model

We represent an image as a collection of detected feature sets that are statistically generated by an instance of our object and camera. Each image feature sets as arising from a corresponding feature generator that depends on projected object information. For this work we generate edge points from projected object contours and image foreground from colored surface points (Figure 3).

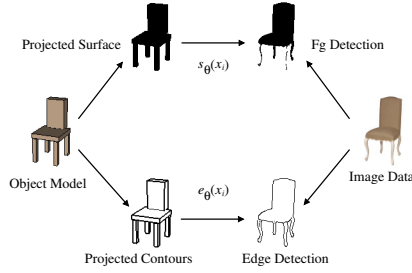

Figure 3: Example of the generative image model for detected features. The left side of the figure gives a rendering of the object and camera models fit to the image on the right side. The rightward arrows show the process of statistical generation of image features. The leftward arrows are feature detection in the image data.

We assume that feature responses are conditionally independent given the model and that the $G$ different types of features are also independent. Denoting the detected feature sets in the $d^{\text{th}}$ image by $\mathbf{x}_d = \mathbf{x}_{d1}, \ldots, \mathbf{x}_{dG}$, we expand the image component of equation (3) to

$$p^{(n_m)}(\mathbf{x}_d \mid \mathbf{c}_d, \mathbf{s}_{md}, \mathbf{t}_m) = \prod_{g=1}^{G} \prod_{i=1}^{N_x} f_{\theta g}^{(n_m)}(x_{dgi}) \, . \tag{5}$$

The function $f_{\theta g}^{(n_m)}(\cdot)$ measures the likelihood of a feature generator producing the response of a detector at each pixel using our object and camera models. Effective construction and implementation of the edge and surface point generators is intricate, and thus we only briefly summarize them. Please refer to our technical report [21] for more details.

**Edge point generator.** We model edge point location and orientation as generated from projected 3D contours of our object model. Since the feature generator likelihood in (5) is computed over all detection responses in an image, we define the edge generator likelihood as

$$\prod_{i=1}^{N_x} f_\theta(x_i) = \prod_{i=1}^{N_x} e_\theta(x_i)^{\mathcal{E}_i} \cdot e'_\theta(x_i)^{(1-\mathcal{E}_i)} \, , \tag{6}$$

where the probability density function $e_\theta(\cdot)$ gives the likelihood of detected edge point at the $i^{\text{th}}$ pixel, and $e'_\theta(\cdot)$ is the density for pixel locations not containing an edge point. The indicator $\mathcal{E}_i$ is 1 if the pixel is an edge point and 0 otherwise. This can be approximated by [21]

$$\prod_{i=1}^{N_x} f_\theta(x_i) \approx \left\{ \prod_{i=1}^{N_x} \widetilde{e}_\theta(x_i)^{\mathcal{E}_i} \right\} e_{\text{bg}}^{N_{\text{bg}}} \, e_{\text{miss}}^{N_{\text{miss}}}, \tag{7}$$

where $e_{\text{bg}}^{N_{\text{bg}}}$ and $e_{\text{miss}}^{N_{\text{miss}}}$ are the probabilities of background and missing detections and $N_{\text{bg}}$ and $N_{\text{miss}}$ are the number of background and missing detections. The density $\widetilde{e}_\theta$ approximates $e_\theta$ by estimating the most likely correspondence between observed edge points and model edges.

To compute the edge point density $e_\theta$, we assume correspondence and use the $i^{\text{th}}$ edge point generated from the $j^{\text{th}}$ model point as a Gaussian distributed displacement $d_{ij}$ in the direction perpendicular of the projected model contour. We further define the gradient direction of the generated edge point to have Gaussian error in its angle difference $\phi_{ij}$ with the perpendicular direction of the projected contour. If $m_j$ is a the model point assumed to generate $x_i$, then

$$e_\theta(x_i) = c_e \, \mathcal{N}\left(d_{ij}; 0, \sigma_d\right) \mathcal{N}\left(\phi_{ij}; 0, \sigma_\phi\right) \tag{8}$$

where the perpendicular distance between $x_i$ and $m_j$ and angular difference between edge point gradient $\mathbf{g}_i$ and model contour perpendicular $\mathbf{v}_j$ are defined $d_{ij} = \| x_i - m_j \|$ and $\phi_{ij} = \cos^{-1}\left(\mathbf{g}_i^{\mathsf{T}} \mathbf{v}_j / \|\mathbf{g}_i\| \, \|\mathbf{v}_j\|\right)$. The range of $d_{ij}$ is $\geq 0$, and the angle $\phi_{ij}$ is in $[0, 1]$.

**Surface point generator.** Surface points are the projected points of viewable surfaces in our object model. Image foreground pixels are found using $k$-means clustering on pixel intensities. Setting $k = 2$ works well as our training images were selected to have minimal clutter. Surface point detections intersecting with model surface projection leads to four easily identifiable cases: foreground, background, missing, and noise. Similar to the edge point generator, the surface point generator likelihood expands to

$$\prod_{i=1}^{N_x} f_\theta(x_i) = s_{\text{fg}}^{N_{\text{fg}}} \, s_{\text{bg}}^{N_{\text{bg}}} \, s_{\text{noise}}^{N_{\text{noise}}} \, s_{\text{miss}}^{N_{\text{miss}}}, \tag{9}$$

# 3 Learning

To learn a category model, we sample the posterior, $p\left(\mathbf{\Theta}^{(\mathbf{n})} \mid \mathbf{X}\right) \propto p\left(\mathbf{X}, \mathbf{\Theta}^{(\mathbf{n})}\right)$, to find good parameters shared by images of multiple object examples from the category. Given enough iterations, a good sampler converges to the target distribution and an optimal value can be readily discovered in the process. However, our posterior distribution is highly convoluted with many sharp, narrow ridges for close fits to the edge points and foreground. In our domain, as in many similar problems, standard sampling techniques tend to get trapped in these local extrema for long periods of time. Our strategy for inference is to combine a mixture of sampling techniques with different strengths in exploring the posterior distribution while still maintaining convergence conditions.

Our sampling space is over all category and instance parameters for a set of input images. We denote the space over an instance of the camera and object models with $n$ parts as $\mathbf{C} \times \mathbf{S}^{(n)}$. Let $\mathbf{T}^{(\mathbf{n})}$ be the space over all topologies and $\mathbf{R}_{\mathbf{c}}^{(\mathbf{n})} \times \mathbf{R}_{\mathbf{s}}^{(\mathbf{n})}$ over all category statistics. The complete sampling space with $m$ subcategories and $D$ instances is then defined as

$$\Omega = \bigcup_{\mathbf{n} \in \mathbb{N}^m} \mathbf{C}^D \times \mathbf{S}^{(\mathbf{n})D} \times \mathbf{T}^{(\mathbf{n})} \times \mathbf{R}_{\mathbf{c}}^{(\mathbf{n})} \times \mathbf{R}_{\mathbf{s}}^{(\mathbf{n})}, \tag{10}$$

Our goal is to sample the posterior with $\mathbf{\Theta}^{(\mathbf{n})} \in \Omega$ such that we find the set of parameters that maximizes it. Since the number of parameters in the sampling space is a unknown, some proposals must change the model dimension. In particular, these *jump moves* (following the terminology of Tu and Zhu [27]) arise from changes in topology. *Diffusion moves* make changes to parameters within a given topology. We cycle between the two kinds of moves.

**Diffusion moves for sampling within topology.** We found that a multivariate Gaussian with small covariance values on the diagonal to be a good proposal distribution for the instance parameters. Proposals for block size changes are done in one of two ways: scaling or shifting attached blocks. We found that both are useful good exploration of the object structure parameter space. Category parameters were sampled by making proposals from the Gamma priors.

Using standard Metropolis-Hastings (MH) [1, 14], the proposed moves are accepted with probability

$$\alpha\left(\tilde{\boldsymbol{\theta}}^{(\mathbf{n})}\right) = \min\left\{1, \frac{p(\tilde{\boldsymbol{\theta}}^{(\mathbf{n})} \mid \mathbf{X})\, q(\boldsymbol{\theta}^{(\mathbf{n})} \mid \tilde{\boldsymbol{\theta}}^{(\mathbf{n})})}{p(\boldsymbol{\theta}^{(\mathbf{n})} \mid \mathbf{X})\, q(\tilde{\boldsymbol{\theta}}^{(\mathbf{n})} \mid \boldsymbol{\theta}^{(\mathbf{n})})}\right\}. \tag{11}$$

The MH diffusion moves exhibit a random walk behavior and can take extended periods of time with many rejections to converge and properly mix well in regions of high probability in the target distribution. Hence we occasionally follow a hybrid Markov chain based on stochastic dynamics, where our joint density is used in a potential energy function. We use the common leapfrog discretization [18] to follow the dynamics and sample from phase space. The necessary derivative calculations are approximated using numerical differentiation (details in [21]).

**Jump moves for topology changes.** For jump moves, we use the trans-dimensional sampling approach outlined by Green [7]. For example, in the case of a block birth in the model, we modify the standard MH acceptance probability to

$$\alpha\left(\tilde{\boldsymbol{\theta}}^{(\mathbf{n}+1)}\right) = \min\left\{1, \frac{p(\tilde{\boldsymbol{\theta}}^{(\mathbf{n}+1)} \mid \mathbf{X})}{p(\boldsymbol{\theta}^{(\mathbf{n})} \mid \mathbf{X})\, q(\tilde{\mathbf{b}}, \tilde{\mathbf{t}})} \frac{r_d}{r_b} \left| \frac{\partial(\tilde{\boldsymbol{\theta}}^{(\mathbf{n}+1)})}{\partial(\boldsymbol{\theta}^{(\mathbf{n})}, \tilde{\mathbf{b}}, \tilde{\mathbf{t}})} \right| \right\}. \tag{12}$$

The jump proposal distribution generates a new block and attachment edge in the topology that are directly used in the proposed object model. Hence, the change of variable factor in the Jacobian reduces to 1. The probability of selecting a birth move versus a death move is given by the ratio of $r_d/r_b$, which we have also defined to be 1. The complimentary block death move is similar with the inverse ratio of posterior and proposal distributions. We additionally define split and merge moves. These are essential moves in our case because the sampler often generates blocks with strong partial fits and proposing splitting it is often accepted.

# 4 Results

We evaluated our model and its inference with image sets of furniture categories, including tables, chairs, sofas, footstools, and desks. We have 30 images in each category containing a single arbitrary

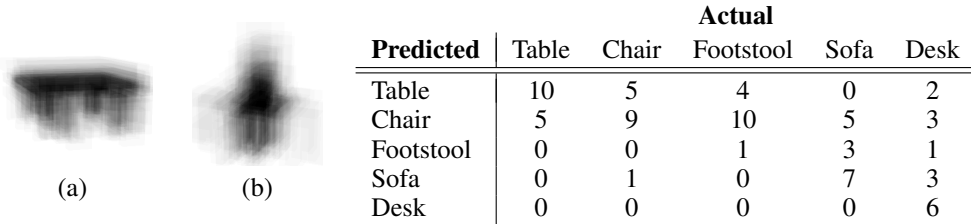

|  | **Actual** | | | | |
| **Predicted** | Table | Chair | Footstool | Sofa | Desk |
| --- | --- | --- | --- | --- | --- |
| Table | 10 | 5 | 4 | 0 | 2 |
| Chair | 5 | 9 | 10 | 5 | 3 |
| Footstool | 0 | 0 | 1 | 3 | 1 |
| Sofa | 0 | 1 | 0 | 7 | 3 |
| Desk | 0 | 0 | 0 | 0 | 6 |

Figure 4: Generated samples of tables (a) and chairs (b) from the learned structure topology and statistical category parameters. The table shows the confusion matrix for object category recognition.

view of the object instance. The images we selected for our data set have the furniture object prominently in the foreground. This enables focusing on evaluating how well we learn 3D structure models of objects.

Inference of the object and camera instances was done on detected edge and surface points in the images. We applied a Canny-based detector for the edges in each image, using the same parameterization each time. Thus, the images contain some edge points considered noise or that are missing from obvious contours. To extract the foreground, we applied a dynamic-threshold discovered in each image with a $k$-means algorithm. Since the furniture objects in the images primarily occupy the image foreground, the detection is quite effective.

We learned the object structure for each category over a 15-image subset of our data for training purposes. We initialized each run of the sampler with a random draw of the category and instance parameters. This is accomplished by first sampling the prior for the object position, rotation and camera view; initially there are no structural elements in the model. We then sample the likelihoods for the instance parameters. The reversible-jump moves in the sampler iteratively propose adding and removing object constructs to the model. The mixture of moves in the sampler was 1-to-1 for jump and diffusion and very infrequently performing a stochastic dynamics chain. Figure 6 shows examples of learned furniture categories and their instances to images after 100K iterations. We visualize the inferred structure topology and statistics in Figure 4 with generated samples from the learned table and chair categories. We observe that the topology of the object structure is quickly established after roughly 10K iterations, this can be seen in Figure 5, which shows the simultaneous inference of two table instances through roughly 10K iterations.

We tested the recognition ability of the learned models on a held out 15-image subset of our data for each category. For each image, we draw a random sample from the category statistics and a topology and begin the diffusion sampling process to fit it. The best overall fit according to the joint density is declared the predicted category. The confusion matrix shown in Figure 4 shows mixed results. Overall, recognition is substantively better than chance (20%), but we expect that much better results are possible with our approach. We conclude from the learned models and confusion matrix that the chair topology shares much of its structure with the other categories and causes the most mistakes. We continue to experiment with larger training data sets, clustering category structure, and longer run times to get better structure fits in the difficult training examples, each of which could help resolve this confusion.

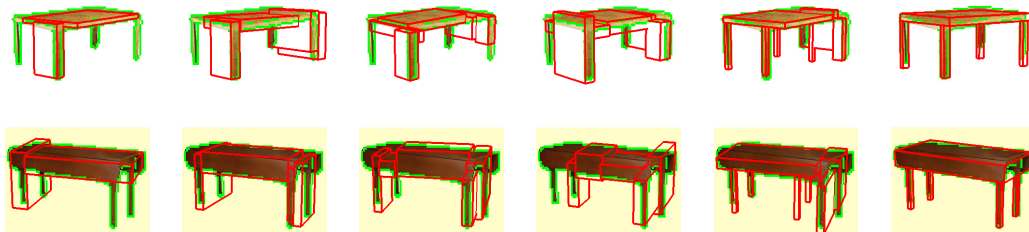

Figure 5: From left to right, successive random samples from 2 of 15 table instances, each after 2K iterations of model inference. The category topology and statistics are learned simultaneously from the set of images; the form of the structure is shared across instances.

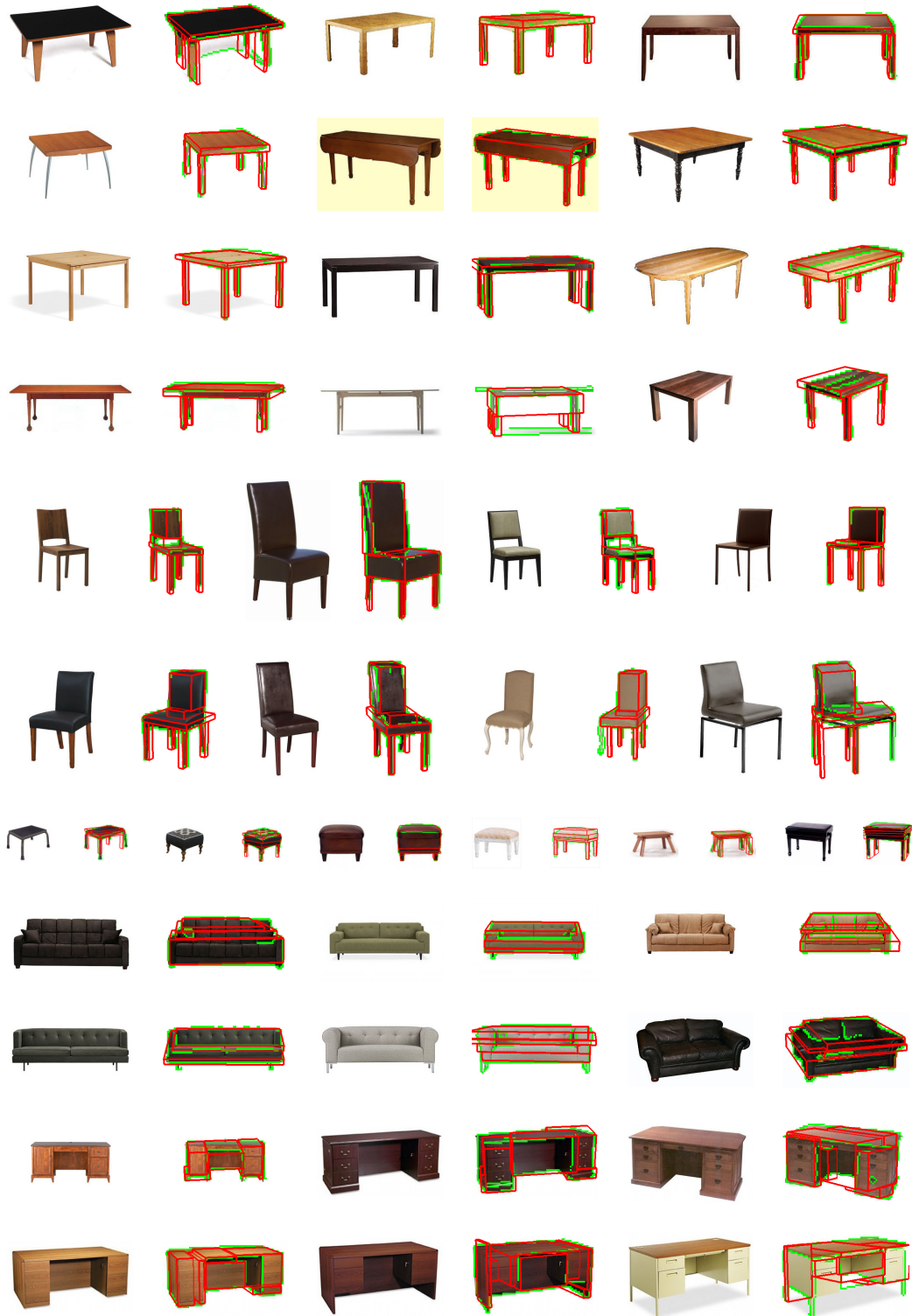

Figure 6: Learning the topology of furniture objects. Sets of contiguous blocks were fit across five image data sets. Model fitting is done jointly for the fifteen images of each set. The fits for the training examples is shown by the blocks drawn in red. Detected edge points are shown in green.

## Acknowledgments

This work is supported in part by NSF CAREER Grant IIS-0747511.

# References

[1] C. Andrieu, N. de Freitas, A. Doucet, and M. I. Jordan. An introduction to MCMC for machine learning. *Machine Learning*, 50(1):5–43, 2003.

[2] I. Biederman. Recognition-by-components: A theory of human image understanding. *Psychological Review*, 94(2):115–147, April 1987.

[3] M. B. Clowes. On seeing things. *Artificial Intelligence*, 2(1):79–116, 1971.

[4] D. Crandall and D. Huttenlocher. Weakly-supervised learning of part-based spatial models for visual object recognition. In *9th European Conference on Computer Vision*, 2006.

[5] L. Fei-Fei, R. Fergus, and P. Perona. Learning generative visual models from few training examples: an incremental bayesian approach tested on 101 object categories. In *Workshop on Generative-Model Based Vision*, 2004.

[6] R. Fergus, P. Perona, and A. Zisserman. Object class recognition by unsupervised scale-invariant learning. In *IEEE Conference on Computer Vision and Pattern Recognition*, 2003.

[7] P. J. Green. Reversible jump Markov chain Monte Carlo computation and Bayesian model determination. *Biometrika*, 82(4):711–732, 1995.

[8] P. J. Green. Trans-dimensional markov chain monte carlo. In *Highly Structured Stochastic Systems*. 2003.

[9] D. Hoiem, C. Rother, and J. Winn. 3d layoutcrf for multi-view object class recognition and segmentation. In *CVPR*, 2007.

[10] D. Huttenlocher and S. Ullman. Recognizing solid objects by alignment with an image. *IJCV*, 5(2):195–212, 1990.

[11] C. Kemp and J. B. Tenenbaum. The discovery of structural form. *Proceedings of the National Academy of Sciences*, 105(31):10687–10692, 2008.

[12] A. Kushal, C. Schmid, and J. Ponce. Flexible object models for category-level 3d object recognition. In *CVPR*, 2007.

[13] M. Leordeanu, M. Hebert, and R. Sukthankar. Beyond local appearance: Category recognition from pairwise interactions of simple features. In *CVPR*, 2007.

[14] J. S. Liu. *Monte Carlo Strategies in Scientific Computing*. Springer-Verlag, 2001.

[15] D. G. Lowe. Fitting parameterized three-dimensional models to images. *IEEE Transactions on Pattern Analysis and Machine Intelligence*, 13(5):441–450, 1991.

[16] D. G. Lowe. Distinctive image features from scale-invariant keypoint. *International Journal of Computer Vision*, 60(2):91–110, 2004.

[17] G. Mori and J. Malik. Recovering 3d human body configurations using shape contexts. *IEEE Transactions on Pattern Analysis and Machine Intelligence*, 2006.

[18] R. M. Neal. Probabilistic inference using Markov chain Monte Carlo methods. Technical Report CRG-TR-93-1, University of Toronto, 1993.

[19] S. Savarese and L. Fei-Fei. 3d generic object categorization, localization and pose estimation. In *IEEE Intern. Conf. in Computer Vision (ICCV)*, 2007.

[20] S. Savarese and L. Fei-Fei. View synthesis for recognizing unseen poses of object classes. In *European Conference on Computer Vision (ECCV)*, 2008.

[21] J. Schlecht and K. Barnard. Learning models of object structure. Technical report, University of Arizona, 2009.

[22] C. Sminchisescu. Kinematic jump processes for monocular 3d human tracking. In *Computer vision and pattern recognition*, 2003.

[23] C. Sminchisescu and B. Triggs. Estimating articulated human motion with covariance scaled sampling. *International Journal of Robotics Research*, 22(6):371–393, 2003.

[24] E. B. Sudderth, A. Torralba, W. T. Freeman, and A. S. Willsky. Learning hierarchical models of scenes, objects, and parts. In *ICCV*, 2005.

[25] K. Sugihara. A necessary and sufficient condition for a picture to represent a polyhedral scene. *IEEE Transactions on Pattern Analysis and Machine Intelligence*, 6(5):578–586, September 1984.

[26] J. B. Tenenbaum, T. L. Griffiths, and C. Kemp. Theory-based bayesian models of inductive learning and reasoning. *Trends in Cognitive Sciences*, 10(7):309–318, 2006.

[27] Z. Tu and S.-C. Zhu. Image segmentation by data-driven markov chain monte-carlo. *IEEE Trans. Patt. Analy. Mach. Intell.*, 24(5):657–673, 2002.

[28] P. H. Winston. Learning structural descriptions from examples. In P. H. Winston, editor, *The psychology of computer vision*, pages 157–209. McGraw-Hill, 1975.

[29] L. Zhu, Y. Chen, and A. Yuille. Unsupervised learning of a probabilistic grammar for object detection and parsing. In *NIPS*, 2006.

[30] S. Zhu and D. Mumford. A stochastic grammar of images. *Foundations and Trends in Computer Graphics and Vision*, 4(2):259–362, 2006.

